# Nets with Unreliable Hidden Nodes Learn Error-Correcting Codes

**Stephen Judd**
Siemens Corporate Research
755 College Road East
Princeton NJ 08540
*judd@learning.siemens.com*

**Paul W. Munro**
Department of Information Science
University of Pittsburgh
Pittsburgh, PA 15260
*munro@lis.pitt.edu*

## ABSTRACT

In a multi-layered neural network, any one of the hidden layers can be viewed as computing a distributed representation of the input. Several "encoder" experiments have shown that when the representation space is small it can be fully used. But computing with such a representation requires completely dependable nodes. In the case where the hidden nodes are noisy and unreliable, we find that error correcting schemes emerge simply by using noisy units during training; random errors injected during backpropagation result in spreading representations apart. Average and minimum distances increase with misfire probability, as predicted by coding-theoretic considerations. Furthermore, the effect of this noise is to protect the machine against permanent node failure, thereby potentially extending the useful lifetime of the machine.

## 1 INTRODUCTION

The encoder task described by Ackley, Hinton, and Sejnowski (1985) for the Boltzmann machine, and by Rumelhart, Hinton, and Williams (1986) for feed-forward networks, has been used as one of several standard benchmarks in the neural network literature. Cottrell, Munro, and Zipser (1987) demonstrated the potential of such autoencoding architectures to lossy compression of image data. In the encoder architecture, the weights connecting the input layer to the hidden layer play the role of an encoding mechanism, and the hidden-output weights are analogous to a decoding device. In the terminology of Shannon and Weaver (1949), the hidden layer corresponds to the communication channel. By analogy, channel noise corresponds to a fault (misfiring) in the hidden layer. Previous

encoder studies have shown that the representations in the hidden layer correspond to opti-
mally efficient (i.e., fully compressed) codes, which suggests that introducing noise in
the form of random interference with hidden unit function may lead to the development of
codes more robust to noise of the kind that prevailed during learning. Many of these
ideas also appear in Chiueh and Goodman (1987) and Séquin and Clay (1990).

We have tested this conjecture empirically, and analyzed the resulting solutions, using a
standard gradient-descent procedure (backpropagation). Although there are alternative tech-
niques to encourage fault tolerance through construction of specialized error functions
(eg., Chauvin, 1989) or direct attacks (eg., Neti, Schneider, and Young, 1990), we have
used a minimalist approach that simply introduces intermittent node misfirings during
training that mimic the errors anticipated during normal performance.

In traditional approaches to developing error-correcting codes (eg., Hamming, 1980), each
symbol from a source alphabet is mapped to a *codeword* (a sequence of symbols from a
*code alphabet*); the distance between codewords is directly related to the code's robustness.

## 2  METHODOLOGY

Computer simulations were performed using strictly layered feed forward networks. The
nodes of one of the hidden layers randomly misfire during training; in most experiments,
this "channel" layer was the sole hidden layer. Each input node corresponds to a transmit-
ted symbol, output nodes to received symbols, channel representations to codewords;
other layers are introduced as needed to enable nonlinear encoding and/or decoding. After
training, the networks were analyzed under various conditions, in terms of performance
and coding-theoretic measures, such as Hamming distance between codewords.

The response, $r$, of each unit in the channel layer is computed by passing the weighted
sum, $x$, through the hyperbolic tangent (a sigmoid that ranges from -1 to +1). The re-
sponses of those units randomly designated to misfire are then multiplied by -1 as this is
most comparable with concepts from coding theory for binary channels.[1] The misfire op-
eration influences the course of learning in two ways, since the erroneous information is
both passed on to units further "downstream" in the net, and used as the presynaptic factor
in the synaptic modification rule. Note that the derivative factor in the backpropagation
procedure is unaffected for units using the hyperbolic tangent, since $dr/dx = (1+r)(1-r)/2$.

These misfirings were randomly assigned according to various kinds of probability distri-
butions: independent identically distributed (i.i.d), $k$-of-$n$, correlated across hidden units,
and correlated over the input distribution. The hidden unit representations required to han-
dle uncorrelated noise roughly correspond to Hamming spheres[2], and can be decoded by a

[1] Other possible misfire modes include setting the node's activity to zero (or some other
constant) or randomizing it. The most appropriate mode depends on various factors, in-
cluding the situation to be simulated and the type of analysis to be performed. For exam-
ple, simulating neuronal death in a biological situation may warrant a different failure
mode than simulating failure of an electronic component.

[2] Consider an $n$-bit block code, where each codeword lies on the vertex of an $n$-cube. The
Hamming sphere of radius $k$ is the neighborhood of vertices that differ from the codeword
by a number of bits less than or equal to $k$.

single layer of weights; thus the entire network consists of just three sets of units: source-channel-sink. However, correlated noise generally necessitates additional layers.

All the experiments described below use the encoder task described by Ackley, Hinton, and Sejnowki (1986); that is, the input pattern consists of just one unit active and the others inactive. The task is to activate only the corresponding unit in the output layer. By comparison with coding theory, the input units are thus analogous to symbols to be encoded, and the hidden unit representations are analogous to the codewords.

## 3  RESULTS

### 3.1.  PERFORMANCE

The first experiment supports the claim of Séquin and Clay (1990) that training with faults improves network robustness. Four 8-30-8 encoders were trained with fault probability $p = 0$, 0.05, 0.1, and 0.3 respectively. After training, each network was tested with fault probabilities varying from 0.05 to 1.0. The results show enhanced performance for networks trained with a higher rate of hidden unit misfiring. Figure 1 shows four performance curves (one for each *training* fault probability), each as a function of *test* fault probability.

Interesting convergence properties were also observed; as the training fault probabilty, $p$, was varied from 0 to 0.4, networks converge reliably faster for low nonzero values ($0.05 < p < 0.15$) than they do at $p = 0$.

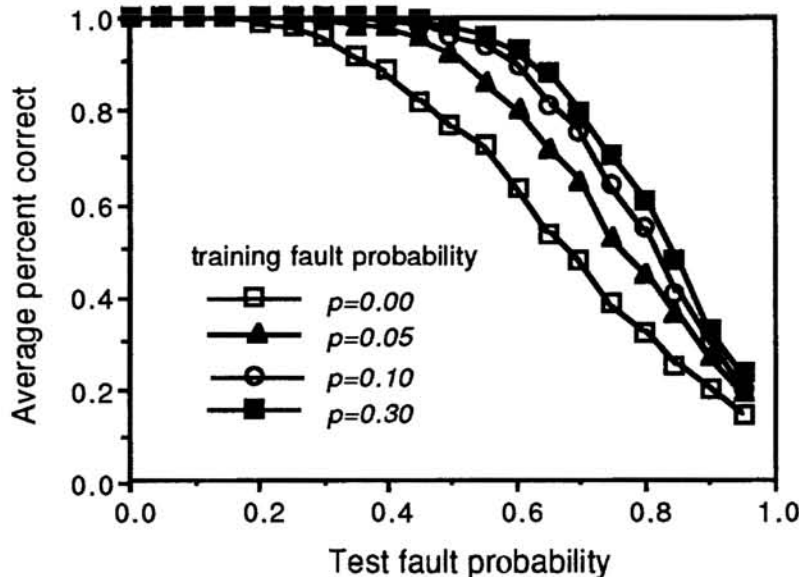

**Figure 1.** *Performance for various training conditions.* Four 8-30-8 encoders were trained with different probabilities for hidden unit misfiring. Each data point is an average over 1000 random stimuli with random hidden unit faults. Outputs are scored correct if the most active output node corresponds to the active input node.

## 3.2.    DISTANCE

### 3.2.1  Distances  increase  with  fault  probability

Distances were measured between all pairs of hidden unit representations. Several networks trained with different fault probabilities and various numbers of hidden units were examined. As expected, both the minimum distances and average distances increase with the training fault probability until it approaches 0.5 per node (see Figure 2). For probabilities above 0.25, the minimum distances fall within the theoretical bounds for a 30 bit code of a 16 symbol alphabet given by Gilbert and Elias (see Blahut, 1987).

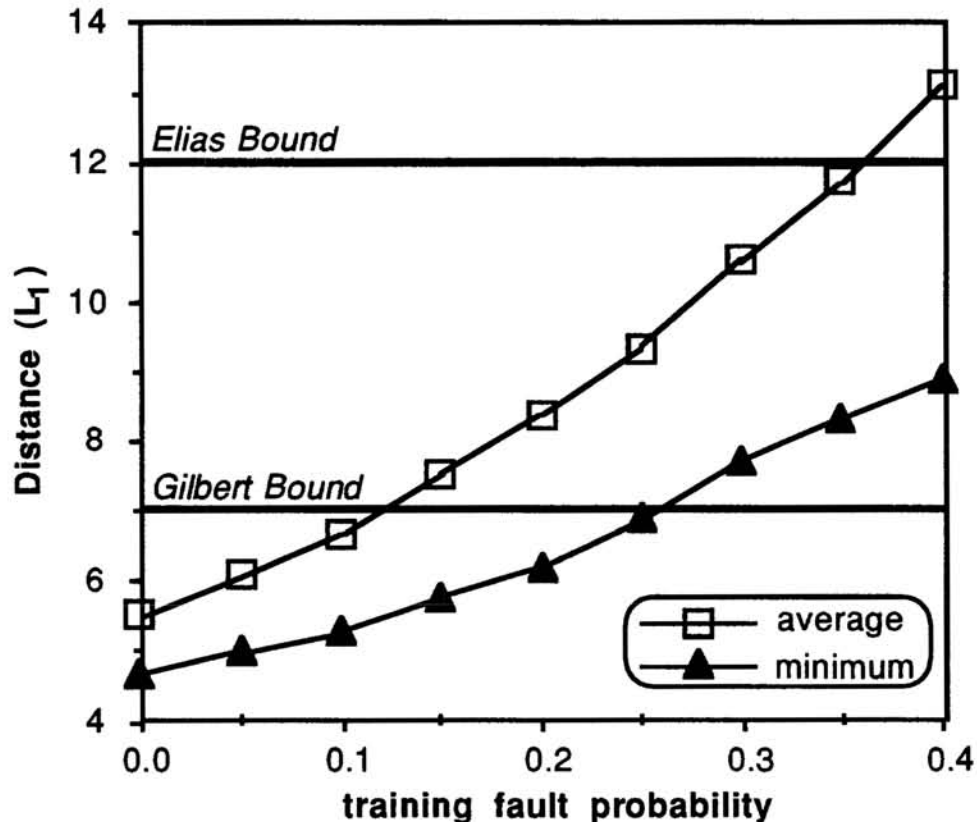

**Figure 2.** *Distance increases with fault probability.* Average and minimum $L_1$ distances are plotted for 16-30-16 networks trained with fault probabilities ranging from 0.0 to 0.4. Each data point represents an average over 100 networks trained using different weight initializations.

### 3.2.2.  Input  probabilities  affect  distance

The probability distribution over the inputs influences the relative distances of the representations at the hidden unit level. To illustrate this, a 4-10-4 encoder was trained using various probabilities for one of the four inputs (denoted P*), distributing the remaining probabilty uniformly among the other three. The average distance between the representation of P* and the others increases with its probability, while the average distance among the other three decreases as shown in the upper part of Figure 3. The more frequent patterns are generally expected to "claim" a larger region of representation space.

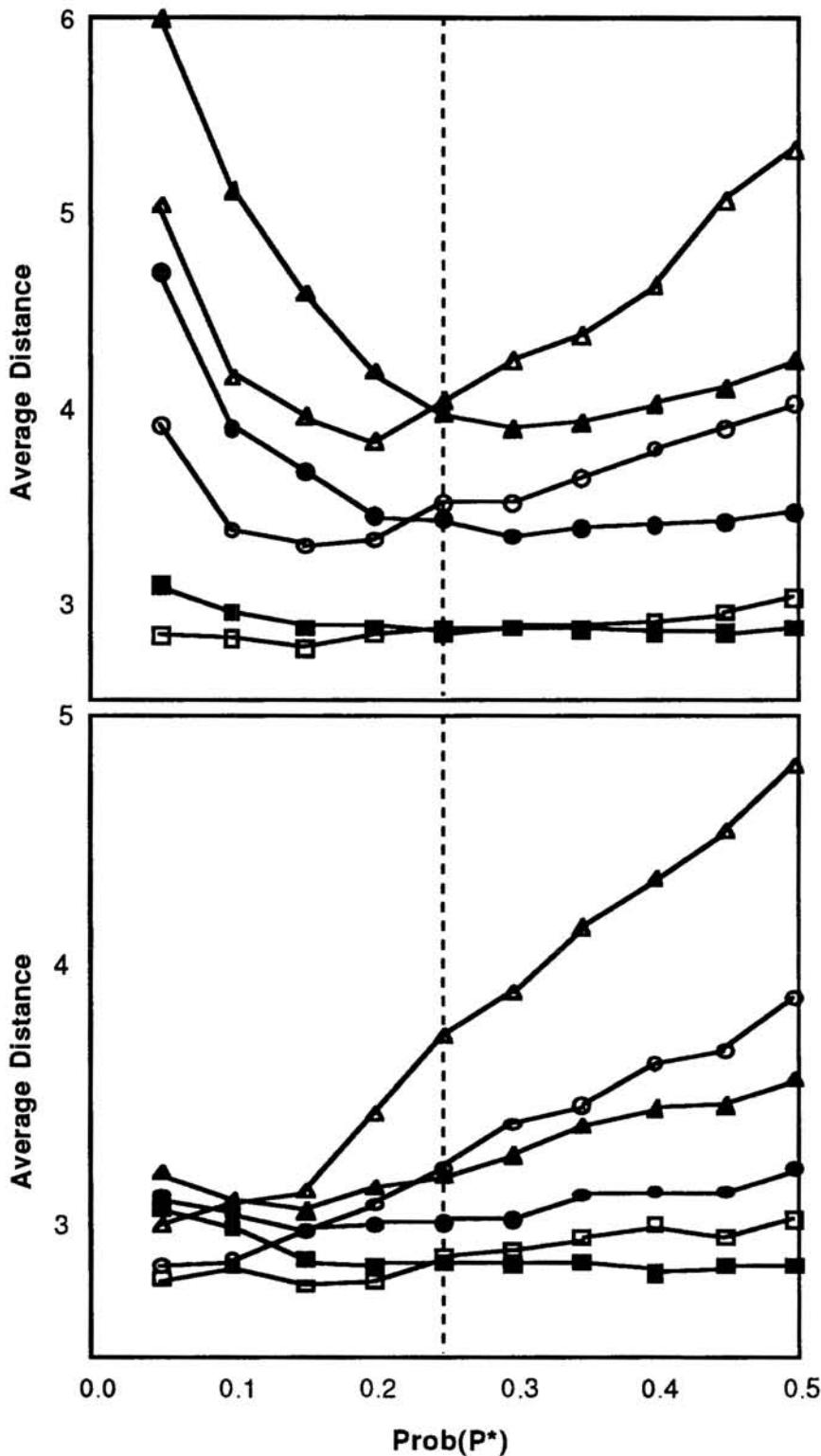

**Figure 3.** *Non-uniform input distribution.* 4-10-4 encoders were trained using failure probabilities of 0 (squares), 0.1 (circles), and 0.2 (triangles). The input distribution was skewed by varying the probability of one of the four items (denoted P*) in the training set from 0.05 to 0.5, keeping the other probabilities uniform. Average $L_1$ distances are shown from the manipulated pattern to the other three (open symbols) and among the equiprobables (filled symbols) as well. In the upper figure, failure is independent of the input, while in the lower figure, failure is induced only when P* is presented.

The dashed line in Figure 3 indicates a uniform input distribution, hence in the top figure, the average distance to P* is equal to the average distances among the other patterns. However this does not hold in the lower figure, indicating that the representations of stimuli that induce more frequent channel errors also claim more representation space.

## 3.3.  CORRELATED MISFIRING

If the error probability for each bit in a message (or each hidden unit in a network layer) is uncorrelated with the other message bits (hidden units), then the principles of distance between codewords (representations) applies.  On the other hand, if there is some structure to the noise (i.e. the misfirings are correlated across the hidden units), there may be different strategies to encoding and decoding, that require computations other than simple distance.  While a Hamming distance criterion on a hypercube is a linearly separable classification function, and hence computable by a single layer of weights, the more general case is not linearly separable, as is demonstrated below.

*Example: Misfiring in 2 of 6 channel units.*
In this example, up to two of six channel units are randomly selected to misfire with each learning trial.  In order to guarantee full recovery from two simultaneous faults, only two symbols can be represented, if the faults are independent; however, if one fault is always in one three-unit subset and the other is always in the complementary subset, it is possible to store four patterns. The following code can be considered with no loss of generality: Let the six hidden units (code bits) be partitioned into two sets of three, where there is at most one fault in each subset.  The four code words, 000000, 000111, 111000, 111111 form an error correcting code under this condition; i.e. each subset is a triplicate code.  Under the allowed fault combinations specified above, any given transmitted code string will be converted by noise to one of 9 strings of the 15 that lie at a Hamming distance of 2 (the 15 unconstrained two-bit errors of the string 000000 are shown in the table below with the 9 that satisfy the constraint in a box).  Because of the symmetric distribution of these 9 allowed states, any category that includes all of them and is defined by a linear (hyperplane) boundary, must include all 15.  Thus, this code cannot be decoded by a single layer of threshold (or sigmoidal) units; hence even if a 4-6-4 network discovers this code, it will not decode it accurately.  However, our experiments show that *inserting a reliable (fault-free) hidden layer of just two units between the channel layer and the output layer (i.e., a 4-6-2-4 encoder)* enables the discovery of a code that is robust to errors of this kind.  The representations of the four patterns in the channel layer show a triply redundant code in each half of the channel layer (Figure 4).  The 2-unit layer provides a transformation that allows successful decoding of channel representations with faults.

## Table.  Possible two-bit error masks

```
000011
000101  000110
 ┌──────────────────────────┐
 │001001  001010  001100    │
 │010001  010010  010100    │ 011000
 │100001  100010  100100    │ 101000  110000
 └──────────────────────────┘
```

Input          Channel          Decoder       Output

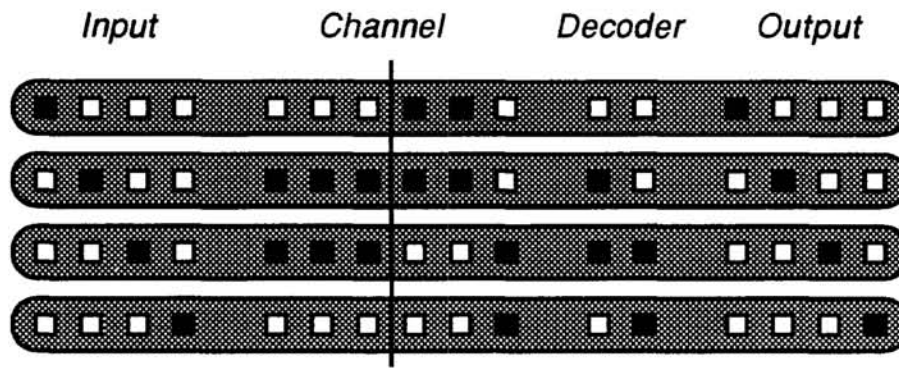

**Figure 4.** *Sample solution to 3-3 channel task.* Thresholded activation patterns are shown for a 4-6-2-4 network. Errors are introduced into the first hidden (channel) layer only. With each iteration, the outputs of one hidden unit from the left half of the hidden layer and one unit from the right half can be inverted. Note that the channel develops a triplicate code for each half-layer.

## 4    DISCUSSION

Results indicate that vanilla backpropagation on its own does not spread out the hidden unit representations (codewords) optimally, and that deliberate random misfiring during training induces wider separations, increasing resistance to node misfiring. Furthermore, non-uniform input distributions and non-uniform channel properties lead to asymmetries among the similarity relationships between hidden unit representations that are consistent with optimizing mutual information.

A mechanism of this kind may be useful for increasing fault tolerance in electronic systems, and may be used in neurobiological systems. The potential usefulness of inducing faults during training extends beyond fault tolerance. Clay and Séquin (1992) point out that training of this kind can enhance the capacity of a network to generalize. In effect, the probability of random faults can be used to vary the number of "effective parameters" (a term coined by Moody, 1992) available for adaptation, without dynamically altering network architecture. Thus, a naive system might begin with a relatively high probability of misfiring, and gradually reduce it as storage capacity needs increase with experience.

This technique may be particularly valuable for designing efficient, robust codes for channels with high order statistical properties, which defy traditional coding techniques. In such cases, a single layer of weights for encoding is not generally sufficient, as was shown above in the 4-6-2-4 example. Additional layers may enhance code efficiency for complex noiseless applications, such as image compression (Cottrell, Munro, and Zipser, 1987).

### Acknowledgements

The second author participated in this research as a visiting research scientist during the summers of 1991 and 1992 at Siemens Corporate Research, which kindly provided financial support and a stimulating research environment.

## References

Ackley, D. H., Hinton, G. E., and Sejnowski, T. J. (1985) A learning algorithm for Boltzmann machines. *Cognitive Science.* **9**: 147-169.

Blahut, R. E. (1987) *Principle and Practise of Information Theory.* Reading MA, Addison Wesley.

Chauvin, Y. (1989) A back-propagation algorithm with optimal use of hidden units. In: Touretsky, D.S. (ed.) *Advances in Neural Information Processing Systems 1.* San Mateo, CA: Morgan Kaufmann Publishers.

Chiueh, Tz-Dar and Rodney Goodman. (1987) A neural network classifier based on coding theory. In: Dana Z. Anderson, editor, *Neural Information Processing Systems*, pp 174--183, New York, A.I.P.

Clay, Reed D. and Séquin, Carlo H. (1992) Fault tolerance training improves generalization and robustness. *Proceedings of IJCNN92*, I-769, Baltimore.

Cottrell, G. W., P. Munro, and D. Zipser (1987) Image compression by back propagation: An example of extensional programming. *Ninth Ann Meeting of the Cognitive Science Society*, pp. 461-473.

Hamming, R. W. (1980) *Coding and Information Theory.* Prentice Hall: Englewood Cliffs, N.J.

Moody, J. (1992) The effective number of parameters. In: Moody, J. E., Hanson, S. J., Lippman, R., (eds.) *Advances in Neural Information Processing Systems 4.* San Mateo, CA: Morgan Kaufmann Publishers.

Neti, C., M. H. Schneider, and E. D. Young. (1990) Maximally fault-tolerant neural networks and nonlinear programming. *Proceedings of IJCNN*, II-483, San Diego.

Rumelhart D., Hinton G., and Williams R. (1986) Learning representations by back-propagating errors. *Nature* 3 2 3:533-536.

Séquin, Carlo H. and Reed D. Clay (1990) Fault tolerance in artificial neural networks. *Proceedings of IJCNN*, I-703, San Diego.

Shannon, C. and Weaver, W. (1949) *The Mathematical Theory of Communication.* University of Illinois Press.
